# Efficient and Robust Feature Extraction by Maximum Margin Criterion

**Haifeng Li**    **Tao Jiang**
Department of Computer Science
University of California
Riverside, CA 92521
{hli,jiang}@cs.ucr.edu

**Keshu Zhang**
Department of Electrical Engineering
University of New Orleans
New Orleans, LA 70148
kzhang1@uno.edu

## Abstract

A new feature extraction criterion, *maximum margin criterion (MMC)*, is proposed in this paper. This new criterion is general in the sense that, when combined with a suitable constraint, it can actually give rise to the most popular feature extractor in the literature, *linear discriminate analysis (LDA)*. We derive a new feature extractor based on MMC using a different constraint that does not depend on the nonsingularity of the within-class scatter matrix $\mathbf{S}_w$. Such a dependence is a major drawback of LDA especially when the sample size is small. The kernelized (nonlinear) counterpart of this linear feature extractor is also established in this paper. Our preliminary experimental results on face images demonstrate that the new feature extractors are efficient and stable.

## 1  Introduction

In statistical pattern recognition, the high-dimensionality is a major cause of the practical limitations of many pattern recognition technologies. In the past several decades, many dimensionality reduction techniques have been proposed. Linear discriminant analysis (LDA, also called Fisher's Linear Discriminant) [1] is one of the most popular linear dimensionality reduction method. In many applications, LDA has been proven to be very powerful. LDA is given by a linear transformation matrix $\mathbf{W} \in \mathcal{R}^{D \times d}$ maximizing the so-called Fisher criterion (a kind of *Rayleigh* coefficient)

$$J_F(\mathbf{W}) = \frac{\mathbf{W}^T \mathbf{S}_b \mathbf{W}}{\mathbf{W}^T \mathbf{S}_w \mathbf{W}} \tag{1}$$

where $\mathbf{S}_b = \sum_{i=1}^{c} p_i (\mathbf{m}_i - \mathbf{m})(\mathbf{m}_i - \mathbf{m})^T$ and $\mathbf{S}_w = \sum_{i=1}^{c} p_i \mathbf{S}_i$ are the between-class scatter matrix and the within-class scatter matrix, respectively; $c$ is the number of classes; $\mathbf{m}_i$ and $p_i$ are the mean vector and *a priori* probability of class $i$, respectively; $\mathbf{m} = \sum_{i=1}^{c} p_i \mathbf{m}_i$ is the overall mean vector; $\mathbf{S}_i$ is the within-class scatter matrix of class $i$; $D$ and $d$ are the dimensionalities of the data before and after the transformation, respectively. To maximize (1), the transformation matrix $\mathbf{W}$ must be constituted by the largest eigenvectors of $\mathbf{S}_w^{-1} \mathbf{S}_b$. The purpose of LDA is to maximize the between-class scatter while simultaneously minimizing the within-class scatter. The two-class LDA has a close connection to optimal linear Bayes classifiers. In the two-class case, the transformation matrix $\mathbf{W}$ is just a vector, which is in the same direction as the discriminant in the corresponding optimal Bayes classifier. However, it has been shown that LDA is suboptimal for multi-class problems [2]. A major drawback of LDA is that it cannot be applied when $\mathbf{S}_w$ is singular due to the *small sample size problem* [3]. The small sample size problem arises

whenever the number of samples is smaller than the dimensionality of samples. For example, a $64 \times 64$ image in a face recognition system has 4096 dimensions, which requires more than 4096 training data to ensure that $\mathbf{S}_w$ is nonsingular. So, LDA is not a stable method in practice when the training data are scarce.

In recent years, many researchers have noticed this problem and tried to overcome the computational difficulty with LDA. Tian *et al.* [4] used the pseudo-inverse matrix $\mathbf{S}_w^+$ instead of the inverse matrix $\mathbf{S}_w^{-1}$. For the same purpose, Hong and Yang [5] tried to add a singular value perturbation to $\mathbf{S}_w$ to make it nonsingular. Neither of these methods are theoretically sound because Fisher's criterion is not valid when $\mathbf{S}_w$ is singular. When $\mathbf{S}_w$ is singular, any positive $\mathbf{S}_b$ makes Fisher's criterion infinitely large. Thus, these naive attempts to calculate the (pseudo or approximate) inverse of $\mathbf{S}_w$ may lead to arbitrary (meaningless) results. Besides, it is also known that an eigenvector could be very sensitive to small perturbation if its corresponding eigenvalue is close to another eigenvalue of the same matrix [6].

In 1992, Liu *et al.* [7] modified Fisher's criterion by using the total scatter matrix $\mathbf{S}_t = \mathbf{S}_b + \mathbf{S}_w$ as the denominator instead of $\mathbf{S}_w$. It has been proven that the modified criterion is exactly equivalent to Fisher's criterion. However, when $\mathbf{S}_w$ is singular, the modified criterion reaches the maximum value (*i.e.*, 1) no matter what the transformation $\mathbf{W}$ is. Such an arbitrary transformation cannot guarantee the maximum class separability unless $\mathbf{W}^T \mathbf{S}_b \mathbf{W}$ is maximized. Besides, this method need still calculate an inverse matrix, which is time consuming. In 2000, Chen *et al.* [8] proposed the LDA+PCA method. When $\mathbf{S}_w$ is of full rank, the LDA+PCA method just calculates the maximum eigenvectors of $\mathbf{S}_t^{-1}\mathbf{S}_b$ to form the transformation matrix. Otherwise, a two-stage procedure is employed. First, the data are transformed into the null space $\mathcal{V}_0$ of $\mathbf{S}_w$. Second, it tries to maximize the between-class scatter in $\mathcal{V}_0$, which is accomplished by performing principal component analysis (PCA) on the between-class scatter matrix in $\mathcal{V}_0$. Although this method solves the small sample size problem, it is obviously suboptimal because it maximizes the between-class scatter in the null space of $\mathbf{S}_w$ instead of the original input space. Besides, the performance of the LDA+PCA method drops significantly when $n - c$ is close to the dimensionality $D$, where $n$ is the number of samples and $c$ is the number of classes. The reason is that the dimensionality of the null space $\mathcal{V}_0$ is too small in this situation, and too much information is lost when we try to extract the discriminant vectors in $\mathcal{V}_0$. LDA+PCA also need calculate the rank of $\mathbf{S}_w$, which is an ill-defined operation due to floating-point imprecisions. At last, this method is complicated and slow because too much calculation is involved.

Kernel Fisher's Discriminant (KFD) [9] is a well-known nonlinear extension to LDA. The instability problem is more severe for KFD because $\mathbf{S}_w$ in the (nonlinear) feature space $\mathcal{F}$ is always singular (the rank of $\mathbf{S}_w$ is $n - c$). Similar to [5], KFD simply adds a perturbation $\mu I$ to $\mathbf{S}_w$. Of course, it has the same stability problem as that in [5] because eigenvectors are sensitive to small perturbation. Although the authors also argued that this perturbation acts as some kind of regularization, *i.e.*, a capacity control in $\mathcal{F}$, the real influence in this setting of regularization is not yet fully understood. Besides, it is hard to determine an optimal $\mu$ since there are no theoretical guidelines.

In this paper, a simpler, more efficient, and stable method is proposed to calculate the most discriminant vectors based on a new feature extraction criterion, the *maximum margin criterion (MMC)*. Based on MMC, new linear and nonlinear feature extractors are established. It can be shown that MMC represents class separability better than PCA. As a connection to Fisher's criterion, we may also derive LDA from MMC by incorporating some suitable constraint. On the other hand, the new feature extractors derived above (based on MMC) do not suffer from the small sample size problem, which is known to cause serious stability problems for LDA (based on Fisher's criterion). Different from LDA+PCA, the new feature extractors based on MMC maximize the between-class scatter in the input space instead of the null space of $\mathbf{S}_w$. Hence, it has a better overall performance than LDA+PCA, as confirmed by our preliminary experimental results.

## 2  Maximum Margin Criterion

Suppose that we are given empirical data

$$(x_1, y_1), \ldots, (x_n, y_n) \in \mathcal{X} \times \{\mathcal{C}_1, \ldots, \mathcal{C}_c\}$$

Here, the domain $\mathcal{X} \in \mathcal{R}^D$ is some nonempty set that the patterns $x_i$ are taken from. The $y_i$'s are called labels or targets. By studying these samples, we want to predict the label $y \in \{\mathcal{C}_1, \ldots, \mathcal{C}_c\}$ of some new pattern $x \in \mathcal{X}$. In other words, we choose $y$ such that $(x, y)$ is in some sense similar to the training examples. For this purpose, some measure need be employed to assess similarity or dissimilarity. We want to keep such similarity/dissimilarity information as much as possible after the dimensionality reduction, *i.e.*, transforming $x$ from $\mathcal{R}^D$ to $\mathcal{R}^d$, where $d \ll D$.

If some distance metric is used to measure the dissimilarity, we would hope that a pattern is close to those in the same class but far from those in different classes. So, a good feature extractor should maximize the distances between classes after the transformation. Therefore, we may define the feature extraction criterion as

$$J = \frac{1}{2} \sum_{i=1}^{c} \sum_{j=1}^{c} p_i p_j d(\mathcal{C}_i, \mathcal{C}_j) \tag{2}$$

We call (2) the maximum margin criterion (MMC). It is actually the summation of $\frac{1}{2} c(c-1)$ interclass margins. Like the weighted pairwise Fisher's criteria in [2], one may also define a weighted maximum margin criterion. Due to the page limit, we omit the discussion in this paper.

One may use the distance between mean vectors as the distance between classes, *i.e.*

$$d(\mathcal{C}_i, \mathcal{C}_j) = d(\mathbf{m}_i, \mathbf{m}_j) \tag{3}$$

where $\mathbf{m}_i$ and $\mathbf{m}_j$ are the mean vectors of the class $\mathcal{C}_i$ and the class $\mathcal{C}_j$, respectively. However, (3) is not suitable since it neglects the scatter of classes. Even if the distance between the mean vectors is large, it is not easy to separate two classes that have the large spread and overlap with each other. By considering the scatter of classes, we define the interclass distance (or margin) as

$$d(\mathcal{C}_i, \mathcal{C}_j) = d(\mathbf{m}_i, \mathbf{m}_j) - s(\mathcal{C}_i) - s(\mathcal{C}_j) \tag{4}$$

where $s(\mathcal{C}_i)$ is some measure of the scatter of the class $\mathcal{C}_i$. In statistics, we usually use the generalized variance $|\mathbf{S}_i|$ or overall variance $tr(\mathbf{S}_i)$ to measure the scatter of data. In this paper, we use the overall variance $tr(\mathbf{S}_i)$ because it is easy to analyze. The weakness of the overall variance is that it ignores covariance structure altogether. Note that, by employing the overall/generalized variance, the expression (4) measures the "average margin" between two classes while the minimum margin is used in support vector machines (SVMs) [10].

With (4) and $s(\mathcal{C}_i)$ being $tr(\mathbf{S}_i)$, we may decompose (2) into two parts

$$J = \frac{1}{2} \sum_{i=1}^{c} \sum_{j=1}^{c} p_i p_j (d(\mathbf{m}_i, \mathbf{m}_j) - tr(\mathbf{S}_i) - tr(\mathbf{S}_j))$$

$$= \frac{1}{2} \sum_{i=1}^{c} \sum_{j=1}^{c} p_i p_j d(\mathbf{m}_i, \mathbf{m}_j) - \frac{1}{2} \sum_{i=1}^{c} \sum_{j=1}^{c} p_i p_j (tr(\mathbf{S}_i) + tr(\mathbf{S}_j))$$

The second part is easily simplified to $tr(\mathbf{S}_w)$

$$\frac{1}{2} \sum_{i=1}^{c} \sum_{j=1}^{c} p_i p_j (tr(\mathbf{S}_i) + tr(\mathbf{S}_j)) = \sum_{i=1}^{c} p_i tr(\mathbf{S}_i) = tr\left(\sum_{i=1}^{c} p_i \mathbf{S}_i\right) = tr(\mathbf{S}_w) \tag{5}$$

By employing the Euclidean distance, we may also simplify the first part to $tr(\mathbf{S}_b)$ as follows

$$\frac{1}{2}\sum_{i=1}^{c}\sum_{j=1}^{c}p_ip_jd(\mathbf{m}_i,\mathbf{m}_j)=\frac{1}{2}\sum_{i=1}^{c}\sum_{j=1}^{c}p_ip_j(\mathbf{m}_i-\mathbf{m}_j)^T(\mathbf{m}_i-\mathbf{m}_j)$$

$$=\frac{1}{2}\sum_{i=1}^{c}\sum_{j=1}^{c}p_ip_j(\mathbf{m}_i-\mathbf{m}+\mathbf{m}-\mathbf{m}_j)^T(\mathbf{m}_i-\mathbf{m}+\mathbf{m}-\mathbf{m}_j)$$

After expanding it, we can simplify the above equation to $\sum_{i=1}^{c}p_i(\mathbf{m}_i-\mathbf{m})^T(\mathbf{m}_i-\mathbf{m})$ by using the fact $\sum_{j=1}^{c}p_j(\mathbf{m}-\mathbf{m}_j)=0$. So

$$\frac{1}{2}\sum_{i=1}^{c}\sum_{j=1}^{c}p_ip_jd(\mathbf{m}_i,\mathbf{m}_j)=tr\left(\sum_{i=1}^{c}p_i(\mathbf{m}_i-\mathbf{m})(\mathbf{m}_i-\mathbf{m})^T\right)=tr(\mathbf{S}_b) \qquad (6)$$

Now we obtain

$$J=tr(\mathbf{S}_b-\mathbf{S}_w) \qquad (7)$$

Since $tr(\mathbf{S}_b)$ measures the overall variance of the class mean vectors, a large $tr(\mathbf{S}_b)$ implies that the class mean vectors scatter in a large space. On the other hand, a small $tr(\mathbf{S}_w)$ implies that every class has a small spread. Thus, a large $J$ indicates that patterns are close to each other if they are from the same class but are far from each other if they are from different classes. Thus, this criterion may represent class separability better than PCA. Recall that PCA tries to maximize the total scatter after a linear transformation. But the data set with a large within-class scatter can also have a large total scatter even when it has a small between-class scatter because $\mathbf{S}_t=\mathbf{S}_b+\mathbf{S}_w$. Obviously, such data are not easy to classify. Compared with LDA+PCA, we maximize the between-class scatter in input space rather than the null space of $\mathbf{S}_w$ when $\mathbf{S}_w$ is singular. So, our method can keep more discriminative information than LDA+PCA does.

## 3 Linear Feature Extraction

When performing dimensionality reduction, we want to find a (linear or nonlinear) mapping from the measurement space $\mathcal{M}$ to some feature space $\mathcal{F}$ such that $J$ is maximized after the transformation. In this section, we discuss how to find an optimal linear feature extractor. In the next section, we will generalize it to the nonlinear case.

Consider a linear mapping $\mathbf{W}\in\mathcal{R}^{D\times d}$. We would like to maximize

$$J(\mathbf{W})=tr(\mathbf{S}_b^W-\mathbf{S}_w^W)$$

where $\mathbf{S}_b^W$ and $\mathbf{S}_w^W$ are the between-class scatter matrix and within-class scatter matrix in the feature space $\mathcal{F}$. Since $\mathbf{W}$ is a linear mapping, it is easy to show $\mathbf{S}_b^W=\mathbf{W}^T\mathbf{S}_b\mathbf{W}$ and $\mathbf{S}_w^W=\mathbf{W}^T\mathbf{S}_w\mathbf{W}$. So, we have

$$J(\mathbf{W})=tr\left(\mathbf{W}^T(\mathbf{S}_b-\mathbf{S}_w)\mathbf{W}\right) \qquad (8)$$

In this formulation, we have the freedom to multiply $\mathbf{W}$ with some nonzero constant. Thus, we additionally require that $\mathbf{W}$ is constituted by the unit vectors, *i.e.* $\mathbf{W}=[\mathbf{w}_1\quad\mathbf{w}_2\quad\ldots\quad\mathbf{w}_d]$ and $\mathbf{w}_k^T\mathbf{w}_k=1$. This means that we need solve the following constrained optimization

$$\max\qquad\sum_{k=1}^{d}\mathbf{w}_k^T(\mathbf{S}_b-\mathbf{S}_w)\mathbf{w}_k$$

$$\text{subject to}\quad\mathbf{w}_k^T\mathbf{w}_k-1=0\qquad k=1,\ldots,d$$

Note that, we may also use other constraints in the above. For example, we may require $tr\left(\mathbf{W}^T\mathbf{S}_w\mathbf{W}\right) = 1$ and then maximize $tr\left(\mathbf{W}^T\mathbf{S}_b\mathbf{W}\right)$. It is easy to show that maximizing MMC with such a constraint in fact results in LDA. The only difference is that it involves a constrained optimization whereas the traditional LDA solves an unconstrained optimization. The motivation for using the constraint $\mathbf{w}_k^T\mathbf{w}_k = 1$ is that it allows us to avoid calculating the inverse of $\mathbf{S}_w$ and thus the potential small sample size problem.

To solve the above optimization problem, we may introduce a Lagrangian

$$\mathcal{L}(\mathbf{w}_k,\lambda_k) = \sum_{k=1}^{d}\mathbf{w}_k^T(\mathbf{S}_b-\mathbf{S}_w)\mathbf{w}_k - \lambda_k(\mathbf{w}_k^T\mathbf{w}_k - 1) \tag{9}$$

with multipliers $\lambda_k$. The Lagrangian $\mathcal{L}$ has to be maximized with respect to $\lambda_k$ and $\mathbf{w}_k$. The condition that at the stationary point, the derivatives of $\mathcal{L}$ with respect to $\mathbf{w}_k$ must vanish

$$\frac{\partial\mathcal{L}(\mathbf{w}_k,\lambda_k)}{\partial\mathbf{w}_k} = ((\mathbf{S}_b-\mathbf{S}_w) - \lambda_k\mathbf{I})\mathbf{w}_k = 0 \qquad k = 1,\ldots,d \tag{10}$$

leads to

$$(\mathbf{S}_b-\mathbf{S}_w)\mathbf{w_k} = \lambda_k\mathbf{w}_k \qquad k = 1,\ldots,d \tag{11}$$

which means that the $\lambda_k$'s are the eigenvalues of $\mathbf{S}_b-\mathbf{S}_w$ and the $\mathbf{w}_k$'s are the corresponding eigenvectors. Thus

$$J(\mathbf{W}) = \sum_{k=1}^{d}\mathbf{w}_k^T(\mathbf{S}_b-\mathbf{S}_w)\mathbf{w}_k = \sum_{k=1}^{d}\lambda_k\mathbf{w}_k^T\mathbf{w}_k = \sum_{k=1}^{d}\lambda_k \tag{12}$$

Therefore, $J(\mathbf{W})$ is maximized when $\mathbf{W}$ is composed of the first $d$ largest eigenvectors of $\mathbf{S}_b - \mathbf{S}_w$. Here, we need not calculate the inverse of $\mathbf{S}_w$, which allows us to avoid the small sample size problem easily. We may also require $\mathbf{W}$ to be orthonormal, which may help preserve the shape of the distribution.

## 4 Nonlinear Feature Extraction with Kernel

In this section, we follow the approach of nonlinear SVMs [10] to kernelize the above linear feature extractor. More precisely, we first reformulate the maximum margin criterion in terms of only dot-product $\langle\Phi(\mathbf{x}),\Phi(\mathbf{y})\rangle$ of input patterns. Then we replace the dot-product by some positive definite kernel $k(\mathbf{x},\mathbf{y})$, *e.g.* Gaussian kernel $e^{-\gamma\|x-y\|^2}$.

Consider the maximum margin criterion in the feature space $\mathcal{F}$

$$J^{\Phi}(\mathbf{W}) = \sum_{k=1}^{d}\mathbf{w}_k^T(\mathbf{S}_b^{\Phi}-\mathbf{S}_w^{\Phi})\mathbf{w}_k$$

where $\mathbf{S}_b^{\Phi}$ and $\mathbf{S}_w^{\Phi}$ are the between-class scatter matrix and within-class scatter matrix in $\mathcal{F}$, *i.e.*, $\mathbf{S}_b^{\Phi} = \sum_{i=1}^{c}p_i(\mathbf{m}_i^{\Phi} - \mathbf{m}^{\Phi})(\mathbf{m}_i^{\Phi} - \mathbf{m}^{\Phi})^T$, $\mathbf{S}_w^{\Phi} = \sum_{i=1}^{c}p_i\mathbf{S}_i^{\Phi}$ and $\mathbf{S}_i^{\Phi} = \frac{1}{n_i}\sum_{j=1}^{n_i}(\Phi(\mathbf{x}_j^{(i)}) - \mathbf{m}_i^{\Phi})(\Phi(\mathbf{x}_j^{(i)}) - \mathbf{m}_i^{\Phi})^T$ with $\mathbf{m}_i^{\Phi} = \frac{1}{n_i}\sum_{j=1}^{n_i}\Phi(\mathbf{x}_j^{(i)})$, $\mathbf{m}^{\Phi} = \sum_{i=1}^{c}p_i\mathbf{m}_i^{\Phi}$, and $\mathbf{x}_j^{(i)}$ is the pattern of class $\mathcal{C}_i$ that has $n_i$ samples.

For us, an important fact is that each $\mathbf{w}_k$ lies in the span of $\Phi(\mathbf{x}_1),\Phi(\mathbf{x}_2),\ldots,\Phi(\mathbf{x}_n)$. Therefore, we can find an expansion for $\mathbf{w}_k$ in the form $\mathbf{w}_k = \sum_{l=1}^{n}\alpha_l^{(k)}\Phi(\mathbf{x}_l)$. Using this expansion and the definition of $\mathbf{m}_i^{\Phi}$, we have

$$\mathbf{w}_k^T\mathbf{m}_i^{\Phi} = \sum_{l=1}^{n}\alpha_l^{(k)}\left(\frac{1}{n_i}\sum_{j=1}^{n_i}\langle\Phi(\mathbf{x}_l),\Phi(\mathbf{x}_j^{(i)})\rangle\right)$$

Replacing the dot-product by some kernel function $k(\mathbf{x}, \mathbf{y})$ and defining $(\widetilde{\mathbf{m}}_i)_l = \frac{1}{n_i}\sum_{j=1}^{n_i} k(\mathbf{x}_l, \mathbf{x}_j^{(i)})$, we get $\mathbf{w}_k^T \mathbf{m}_i^\Phi = \boldsymbol{\alpha}_k^T \widetilde{\mathbf{m}}_i$ with $(\boldsymbol{\alpha}_k)_l = \alpha_l^{(k)}$. Similarly, we have

$$\mathbf{w}_k^T \mathbf{m}^\Phi = \mathbf{w}_k^T \sum_{i=1}^c p_i \mathbf{m}_i^\Phi = \boldsymbol{\alpha}_k^T \sum_{i=1}^c p_i \widetilde{\mathbf{m}}_i = \boldsymbol{\alpha}_k^T \widetilde{\mathbf{m}}$$

with $\widetilde{\mathbf{m}} = \sum_{i=1}^c p_i \widetilde{\mathbf{m}}_i$. This means $\mathbf{w}_k^T(\mathbf{m}_i^\Phi - \mathbf{m}^\Phi) = \boldsymbol{\alpha}_k^T(\widetilde{\mathbf{m}}_i - \widetilde{\mathbf{m}})$. and

$$\sum_{k=1}^d \mathbf{w}_k^T \mathbf{S}_b^\Phi \mathbf{w}_k = \sum_{k=1}^d \sum_{i=1}^c p_i (\mathbf{w}_k^T(\mathbf{m}_i^\Phi - \mathbf{m}^\Phi))(\mathbf{w}_k^T(\mathbf{m}_i^\Phi - \mathbf{m}^\Phi))^T$$

$$= \sum_{k=1}^d \sum_{i=1}^c p_i^T \boldsymbol{\alpha}_k^T (\widetilde{\mathbf{m}}_i - \widetilde{\mathbf{m}})(\widetilde{\mathbf{m}}_i - \widetilde{\mathbf{m}})^T \boldsymbol{\alpha}_k = \sum_{k=1}^d \boldsymbol{\alpha}_k^T \widetilde{\mathbf{S}}_b \boldsymbol{\alpha}_k$$

where $\widetilde{\mathbf{S}}_b = \sum_{i=1}^c p_i (\widetilde{\mathbf{m_i}} - \widetilde{\mathbf{m}})(\widetilde{\mathbf{m_i}} - \widetilde{\mathbf{m}})^T$.

Similarly, one can simplify $\mathbf{W}^T \mathbf{S}_w^\Phi \mathbf{W}$. First, we have $\mathbf{w}_k^T(\Phi(\mathbf{x}_j^{(i)}) - \mathbf{m}_i^\Phi) = \boldsymbol{\alpha}_k^T(\mathbf{k}_j^{(i)} - \widetilde{\mathbf{m}}_i)$ with $(\mathbf{k}_j^{(i)})_l = k(\mathbf{x}_l, \mathbf{x}_j^{(i)})$. Considering $\mathbf{w}_k^T \mathbf{S}_i^\Phi \mathbf{w}_k = \frac{1}{n_i}\sum_{j=1}^{n_i}(\mathbf{w}_k^T(\Phi(\mathbf{x}_j^{(i)}) - \mathbf{m}_i^\Phi))(\mathbf{w}_k^T(\Phi(\mathbf{x}_j^{(i)}) - \mathbf{m}_i^\Phi))^T$, we have

$$\mathbf{w}_k^T \mathbf{S}_i^\Phi \mathbf{w}_k = \frac{1}{n_i}\sum_{j=1}^{n_i} \boldsymbol{\alpha}_k^T (\mathbf{k}_j^{(i)} - \widetilde{\mathbf{m}}_i)(\mathbf{k}_j^{(i)} - \widetilde{\mathbf{m}}_i)^T \boldsymbol{\alpha}_k$$

$$= \frac{1}{n_i}\sum_{j=1}^{n_i} \boldsymbol{\alpha}_k^T \widetilde{\mathbf{S}}_i (\mathbf{e}_j - \frac{1}{n_i}\mathbf{1}_{n_i})(\mathbf{e}_j - \frac{1}{n_i}\mathbf{1}_{n_i})^T \widetilde{\mathbf{S}}_i^T \boldsymbol{\alpha}_k$$

$$= \frac{1}{n_i}\sum_{j=1}^{n_i} \boldsymbol{\alpha}_k^T \widetilde{\mathbf{S}}_i (\mathbf{e}_j \mathbf{e}_j^T - \frac{1}{n_i}\mathbf{e}_j \mathbf{1}_{n_i}^T - \frac{1}{n_i}\mathbf{1}_{n_i}\mathbf{e}_j^T + \frac{1}{n_i^2}\mathbf{1}_{n_i}\mathbf{1}_{n_i}^T)\widetilde{\mathbf{S}}_i^T \boldsymbol{\alpha}_k$$

$$= \frac{1}{n_i}\boldsymbol{\alpha}_k^T \widetilde{\mathbf{S}}_i (\mathbf{I}_{n_i \times n_i} - \frac{1}{n_i}\mathbf{1}_{n_i}\mathbf{1}_{n_i}^T)\widetilde{\mathbf{S}}_i^T \boldsymbol{\alpha}_k$$

where $(\widetilde{\mathbf{S}}_i)_{lj} = k(\mathbf{x}_l, \mathbf{x}_j^{(i)})$, $I_{n_i \times n_i}$ is the $n_i \times n_i$ identity matrix, $\mathbf{1}_{n_i}$ is the $n_i$-dimensional vector of 1's, and $\mathbf{e}_j$ is the canonical basis vector of $n_i$ dimensions. Thus, we obtain

$$\sum_{k=1}^d \mathbf{w}_k^T \mathbf{S}_w^\Phi \mathbf{w}_k = \sum_{k=1}^d \sum_{i=1}^c p_i \frac{1}{n_i}\boldsymbol{\alpha}_k^T \widetilde{\mathbf{S}}_i (\mathbf{I}_{n_i} - \frac{1}{n_i}\mathbf{1}_{n_i}\mathbf{1}_{n_i}^T)\widetilde{\mathbf{S}}_i^T \boldsymbol{\alpha}_k$$

$$= \sum_{k=1}^d \boldsymbol{\alpha}_k^T \left( \sum_{i=1}^c p_i \frac{1}{n_i}\widetilde{\mathbf{S}}_i (\mathbf{I}_{n_i} - \frac{1}{n_i}\mathbf{1}_{n_i}\mathbf{1}_{n_i}^T)\widetilde{\mathbf{S}}_i^T \right) \boldsymbol{\alpha}_k = \sum_{k=1}^d \boldsymbol{\alpha}_k^T \widetilde{\mathbf{S}}_w \boldsymbol{\alpha}_k$$

where $\widetilde{\mathbf{S}}_w = \sum_{i=1}^c p_i \frac{1}{n_i}\widetilde{\mathbf{S}}_i (\mathbf{I}_{n_i} - \frac{1}{n_i}\mathbf{1}_{n_i}\mathbf{1}_{n_i}^T)\widetilde{\mathbf{S}}_i^T$. So the maximum criterion in the feature space $\mathcal{F}$ is

$$J(\mathbf{W}) = \sum_{k=1}^d \boldsymbol{\alpha}_k^T (\widetilde{\mathbf{S}}_b - \widetilde{\mathbf{S}}_w)\boldsymbol{\alpha}_k \tag{13}$$

Similar to the observations in Section 3, the above criterion is maximized by the largest eigenvectors of $\widetilde{\mathbf{S}}_b - \widetilde{\mathbf{S}}_w$.

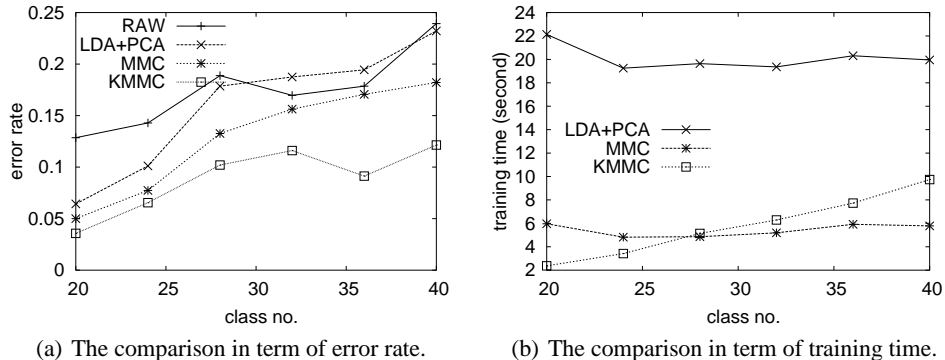

| (a) The comparison in term of error rate. | (b) The comparison in term of training time. |

Figure 1: Experimental results obtained using a linear SVM on the original data (RAW), and the data extracted by LDA+PCA, the linear feature extractor based on MMC (MMC) and the nonlinear feature extractor based on MMC (KMMC), which employs the Gaussian kernel with $\gamma = 0.03125$.

## 5  Experiments

To evaluate the performance of our new methods (both linear and nonlinear feature extractors), we ran both LDA+PCA and our methods on the ORL face dataset [11]. The ORL dataset consists of 10 face images from 40 subjects for a total of 400 images, with some variation in pose, facial expression and details. The resolution of the images is $112 \times 92$, with 256 gray-levels. First, we resized the images to $28 \times 23$ to save the experimental time. Then, we reduced the dimensionality of each image set to $c - 1$, where $c$ is the number of classes. At last we trained and tested a linear SVM on the dimensionality-reduced data. As a control, we also trained and tested a linear SVM on the original data before its dimensionality was reduced.

In order to demonstrate the effectiveness and the efficiency of our methods, we conducted a series of experiments and compared our results with those obtained using LDA+PCA. The error rates are shown in Fig.1(a). When trained with 3 samples and tested with 7 other samples for each class, our method is generally better than LDA+PCA. In fact, our method is usually better than LDA+PCA on other numbers of training samples. To save space, we do not show all the results here. Note that our methods can even achieve lower error rates than a linear SVM on the original data (without dimensionality reduction). However, LDA+PCA does not demonstrate such a clear superiority over RAW. Fig. 1(a) also shows that the kernelized (nonlinear) feature extractor based on MMC is significantly better than the linear one, in particular when the number of classes $c$ is large.

Besides accuracy, our methods are also much more efficient than LDA+PCA in the sense of the training time required. Fig. 1(b) shows that our linear feature extractor is about 4 times faster than LDA+PCA. The same speedup was observed on other numbers of training samples. Note that our nonlinear feature extractor is also faster than LDA+PCA in this case although it is very time-consuming to calculate the kernel matrix in general. An explanation of the speedup is that the kernel matrix size equals the number of samples, which is pretty small in this case.

Furthermore, our method performs much better than LDA+PCA when $n - c$ is close to the dimensionality $D$. Because the amount of training data was limited, we resized the images to 168 dimensions to create such a situation. The experimental results are shown in Fig. 2. In this situation, the performance of LDA+PCA drops significantly because the null space of $\mathbf{S}_w$ has a small dimensionality. When LDA+PCA tries to maximize the between-class scatter in this small null space, it loses a lot of information. On the other hand, our method tries to maximize the between-class scatter in the original input space. From Fig. 2, we can

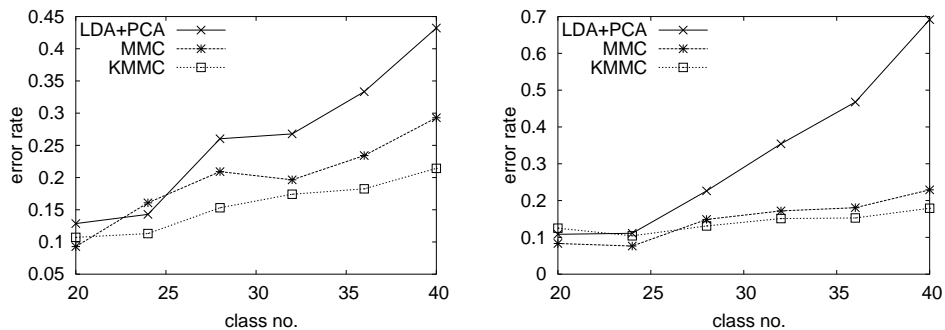

(a) Each class contains three training samples.     (b) Each class contains four training samples.

Figure 2: Comparison between our new methods and LDA+PCA when $n - c$ is close to $D$.

see that LDA+PCA is ineffective in this situation because it is even worse than a random guess. But our method still produced acceptable results. Thus, the experimental results show that our method is better than LDA+PCA in terms of both accuracy and efficiency.

## 6 Conclusion

In this paper, we proposed both linear and nonlinear feature extractors based on the maximum margin criterion. The new methods do not suffer from the small sample size problem. The experimental results show that it is very efficient, accurate, and robust.

### Acknowledgments

We thank D. Gunopulos, C. Domeniconi, and J. Peng for valuable discussions and comments. This work was partially supported by NSF grants CCR-9988353 and ACI-0085910.

## References

[1] R. A. Fisher. The use of multiple measurements in taxonomic problems. *Annual of Eugenics*, 7:179–188, 1936.

[2] M. Loog, R. P. W. Duin, and R. Haeb-Umbach. Multiclass linear dimension reduction by weighted pairwise fisher criteria. *IEEE Transactions on Pattern Analysis and Machine Intelligence*, 23(7):762–766, 2001.

[3] K. Fukunaga. *Introduction to Statistical Pattern Recognition*. Academic Press, New York, 2nd edition, 1990.

[4] Q. Tian, M. Barbero, Z. Gu, and S. Lee. Image classification by the foley-sammon transform. *Optical Engineering*, 25(7):834–840, 1986.

[5] Z. Hong and J. Yang. Optimal discriminant plane for a small number of samples and design method of classifier on the plane. *Pattern Recognition*, 24(4):317–324, 1991.

[6] G. W. Stewart. *Introduction to Matrix Computations*. Academic Press, New York, 1973.

[7] K. Liu, Y. Cheng, and J. Yang. A generalized optimal set of discriminant vectors. *Pattern Recognition*, 25(7):731–739, 1992.

[8] L. Chen, H. Liao, M .Ko, J. Lin, and G. Yu. A new LDA-based face recognition system which can solve the small sample size problem. *Pattern Recognition*, 33(10):1713–1726, 2000.

[9] S. Mika, G. Rätsch, J. Weston, B. Schölkopf, and K.-R. Müller. Fisher discriminant analysis with kernels. In Y.-H. Hu, J. Larsen, E. Wilson, and S. Douglas, editors, *Neural Networks for Signal Processing IX*, pages 41–48. IEEE, 1999.

[10] V. N. Vapnik. *Statistical Learning Theory*. John Wiley & Sons, New York, 1998.

[11] F. Samaria and A. Harter. Parameterisation of a stochastic model for human face identification. In *Proceedings of 2nd IEEE Workshop on Applications of Computer Vision*, Sarasota, FL, 1994.
